# Locomotion in a Lower Vertebrate: Studies of the Cellular Basis of Rhythmogenesis and Oscillator Coupling

James T. Buchanan
Department of Biology
Marquette University
Milwaukee, WI 53233

## Abstract

To test whether the known connectivies of neurons in the lamprey spinal cord are sufficient to account for locomotor rhythmogenesis, a "connectionist" neural network simulation was done using identical cells connected according to experimentally established patterns. It was demonstrated that the network oscillates in a stable manner with the same phase relationships among the neurons as observed in the lamprey. The model was then used to explore coupling between identical oscillators. It was concluded that the neurons can have a dual role as rhythm generators and as coordinators between oscillators to produce the phase relations observed among segmental oscillators during swimming.

## 1 INTRODUCTION

One approach to analyzing neurobiological systems is to use simpler preparations that are amenable to techniques which can investigate the cellular, synaptic, and network levels of organization involved in the generation of behavior. This approach has yielded significant progress in the analysis of rhythm pattern generators in several invertebrate preparations (e.g., the stomatogastric ganglion of lobster, Selverston et al., 1983). We have been carrying out similar types of studies of locomotor rhythm generation in a vertebrate preparation, the lamprey spinal cord, which offers many of the same technical advantages of invertebrate nervous systems. To aid our understanding of how identified lamprey interneurons might participate in rhythmogenesis and in the coupling of oscillators, we have used neural network models.

## 2    FICTIVE SWIMMING

The neuronal correlate of swimming can be induced in the isolated lamprey spinal cord by exposure to glutamate, which is considered to be the principal endogenous excitatory neurotransmitter.  As in the intact swimming lamprey, this "fictive" swimming is characterized by periodic bursts of motoneuron action potentials in

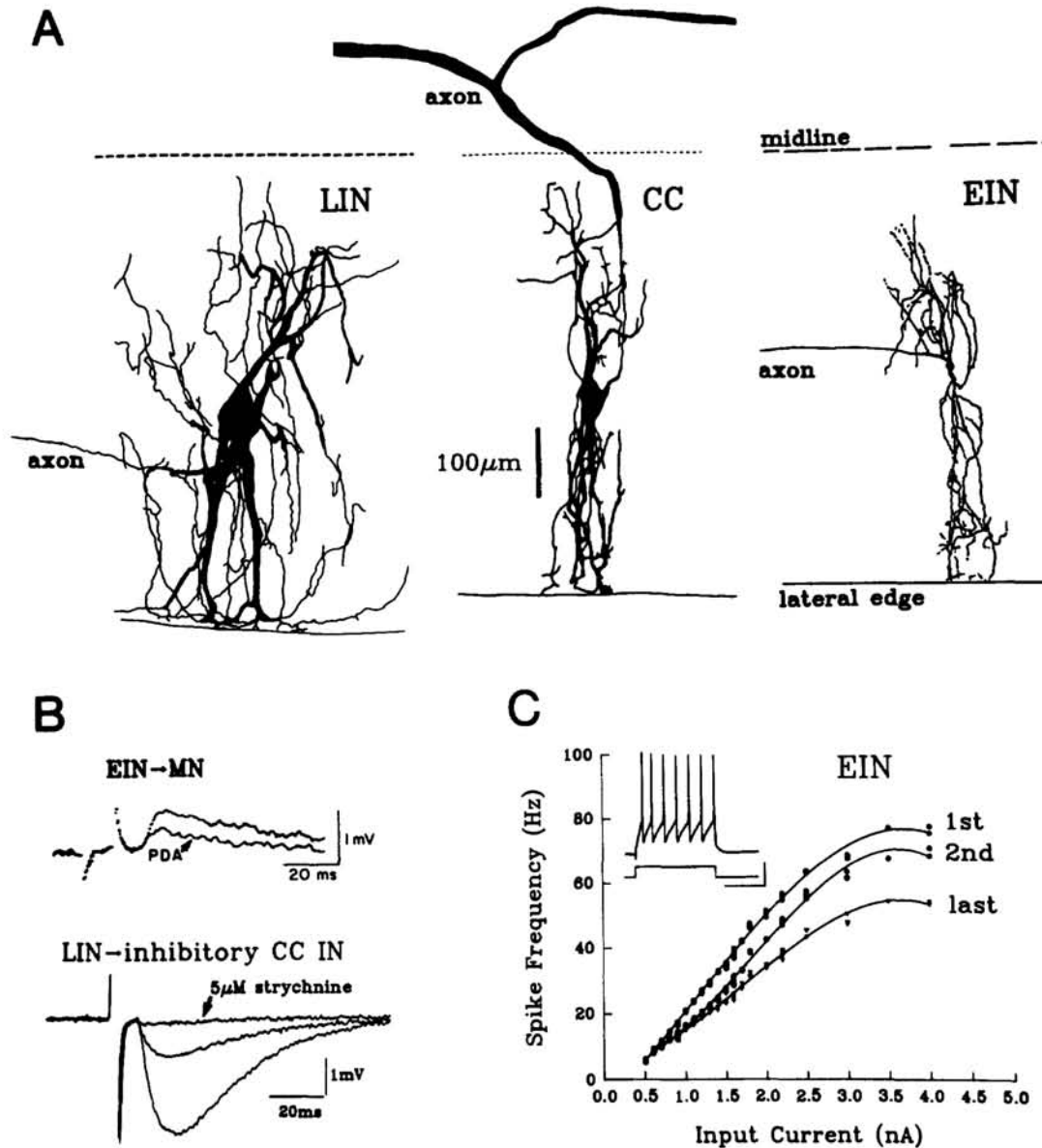

Figure 1: Lamprey spinal interneurons. **A**, drawings of three types of interneurons after intracellular dye injections. **B**, inhibitory and excitatory postsynaptic potentials and the effects of selective antagonists. **C**, firing frequency of the first, second, and last spike intervals during a 400ms current injection.

the ventral roots, and these bursts alternate between sides of the spinal cord and propagate in a head-to-tail direction during forward swimming (Cohen and Wallen, 1980; Wallen and Williams, 1984). Thus, the cellular mechanisms for generating the basic swimming pattern reside within the spinal cord as has been demonstrated for many other vertebrates (Grillner, 1981).

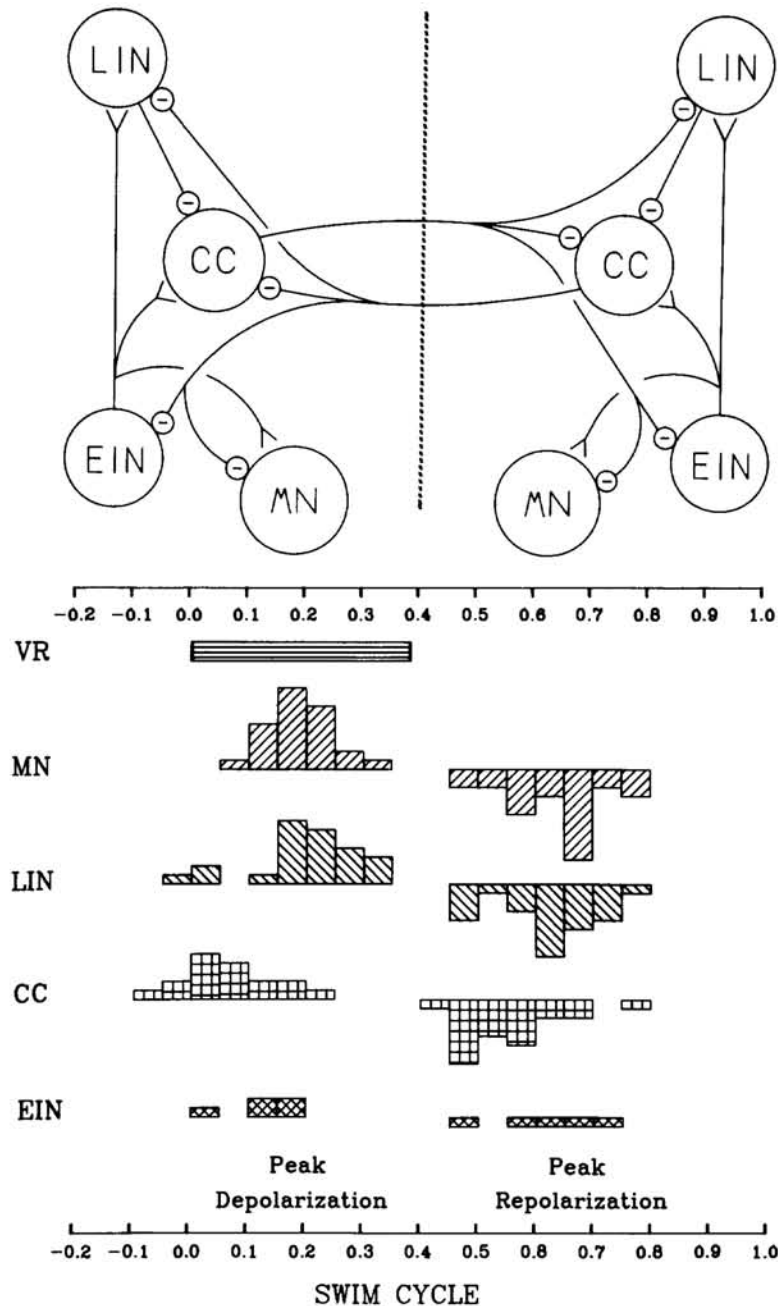

Figure 2: Connectivity and activity patterns. **Top:** synaptic connectivity among the interneurons and motoneurons (MN). **Bottom:** histograms summarizing the activity of cells recorded intracellularly during fictive swimming. Timing of activity of neurons with the onset of the ipsilateral ventral root burst.

The swimming rhythm generator is thought to consist of a chain of coupled oscillators distributed throughout the length of the spinal cord. The isolated spinal cord can be cut into pieces as small as two or three segments in length from any head-to-tail level and still exhibit alternating ventral root bursting upon application of glutamate. The intrinsic swimming frequency in each of these pieces of spinal cord is different by as much as two-fold, and no consistent relationship between intrinsic frequency and the head-to-tail level from which the piece originated has been observed (Cohen, 1986). Thus, coupling among the oscillators must provide some "buffering capacity" to cope with these intrinsic frequency differences. Another feature of the coupling is the constancy of phase lag, such that over a wide range of swimming cycle periods, the delay of ventral root burst onsets between segments is a constant fraction of the cycle period (Wallen and Williams, 1984). Since the cycle period in swimming lamprey can vary over a ten-fold range, axonal conduction time probably is not a factor in the delay between segments.

## 3   SPINAL INTERNEURONS

In recent years, many classes of spinal neurons have been characterized using a variety of neurobiological techniques, particularly intracellular recording of membrane potential (Rovainen, 1974; Buchanan, 1982; Buchanan *et al.*, 1989). Several of these classes of neurons are active during fictive swimming. These include the lateral interneurons (LIN), cells with axons projecting contralaterally and caudally (CC), and the excitatory interneurons (EIN). The LINs are large neurons with an ipsilaterally and caudally projecting inhibitory axon (Fig. 1A,B). The CC interneurons are medium-sized inhibitory cells (Fig. 1A). The EINs are small interneurons with ipsilaterally and either caudally or rostrally projecting axons (Fig. 1A,B,C). The axons of all these cell types project at least five segments and interact with neurons in multiple segments. The neurons have similar resting and firing properties. They are indistinguishable in their resting potentials, their thresholds, and their action potential amplitudes, durations, and after-spike potentials. Their main differences are size-related parameters such as input resistance and membrane time constant. They fire action potentials throughout the duration of long, depolarizing current pulses, showing some adaptation (a declining frequency with successive action potentials). The plots of spike frequency vs. input current for these various cell types are generally monotonic, with a tendency to saturate at higher levels of input current (Fig. 1C)(Buchanan, 1991).

The synaptic connectivites of these cells have been established with simultaneous intracellular recording of pre- and post-synaptic neurons, and the results are summarized in Fig. 2 along with their activity patterns during fictive swimming. All of the cells exhibit oscillating membrane potentials with depolarizing peaks which tend to occur during the ventral root burst and with repolarizing troughs which occur about one-half cycle later (Buchanan and Cohen 1982). These oscillations appear to be due in large part to two phases of synaptic input: an excitatory depolarizing phase and an inhibitory repolarizing phase (Kahn, 1982; Russell and Wallen, 1983). The excitatory phase of motoneurons comes from EINs and the inhibitory phase from CCs. However, these interneurons not only interact with motoneurons but with other interneurons as well. So the possibility exists that these interneurons provide the synaptic drive for all neurons of the network, not just motoneurons. Addition-

ally, it is possible that rhythmicity itself originates from the pattern of synaptic connectivity because the circuit has a basic alternating network of reciprocal inhibition between CC interneurons on opposite sides of the spinal cord. Reciprocal inhibition as an oscillatory network needs some form of burst-termination, and this could be provided by the feedforward inhibition of ipsilateral CC interneurons by the LINs. This inhibition could also account for the early peak observed in many CC interneurons during fictive swimming (Fig. 2).

# 4    NEURAL NETWORK MODEL

The ability of the network of Fig. 2 to generate the basic oscillatory pattern of fictive swimming was tested using a "connectionist" neural network simulation (Buchanan, 1992). All of the cells of the neural network had identical S-shaped input-output curves and differed only in their excitatory levels and their synaptic connectivity, which was set according to the scheme of Fig. 2. If the excitation of CCs was made larger than LINs, the network would oscillate (Fig. 3). These oscillations began fairly promptly and could continued for at least thousands of cycles. The phase relations among the units were similar to those in the lamprey: cells on opposite sides of the spinal cord were anti-phasic while most cells on the same side of the cord were co-active. Significantly, both in the model and in the lamprey, the CCs were phase advanced, presumably due to their inhibition by LINs.

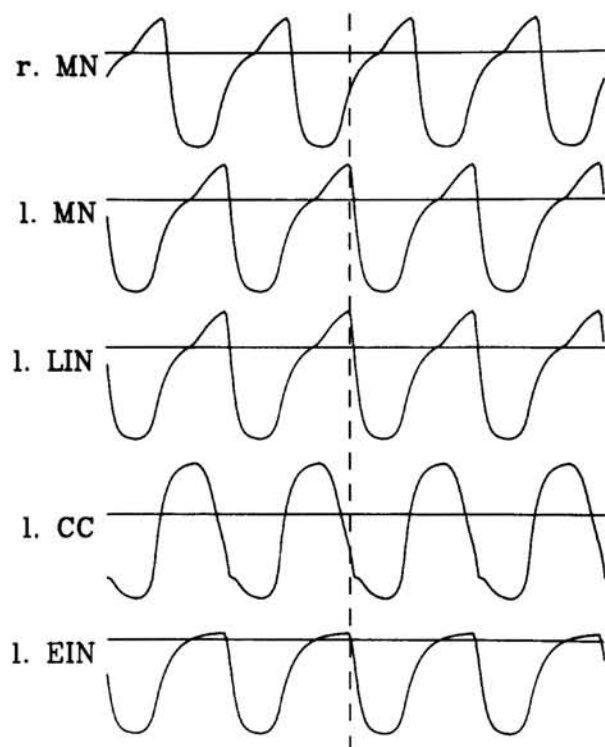

Figure 3: Activity of the neural network model for the lamprey locomotor circuit.

## 4.1  COUPLING

The neural network model of the lamprey swimming oscillator was further used to explore how the coupling among locomotor oscillators might be achieved. Two identical oscillator networks were coupled using the various pairs of cells in one network connected to pairs of cells in the second network. All nine pairs of possible connections were tested since all of the interneurons interact with neurons in multiple segments. The coupling was evaluated by several criteria based on observations of lamprey swimming: 1) the stability of the phase difference between oscillators and the rate of achieving the steady-state, 2) the ability of the coupling to tolerate intrinsic frequency differences between oscillators, and 3) the constancy of the phase lag over a wide range of oscillator frequencies.

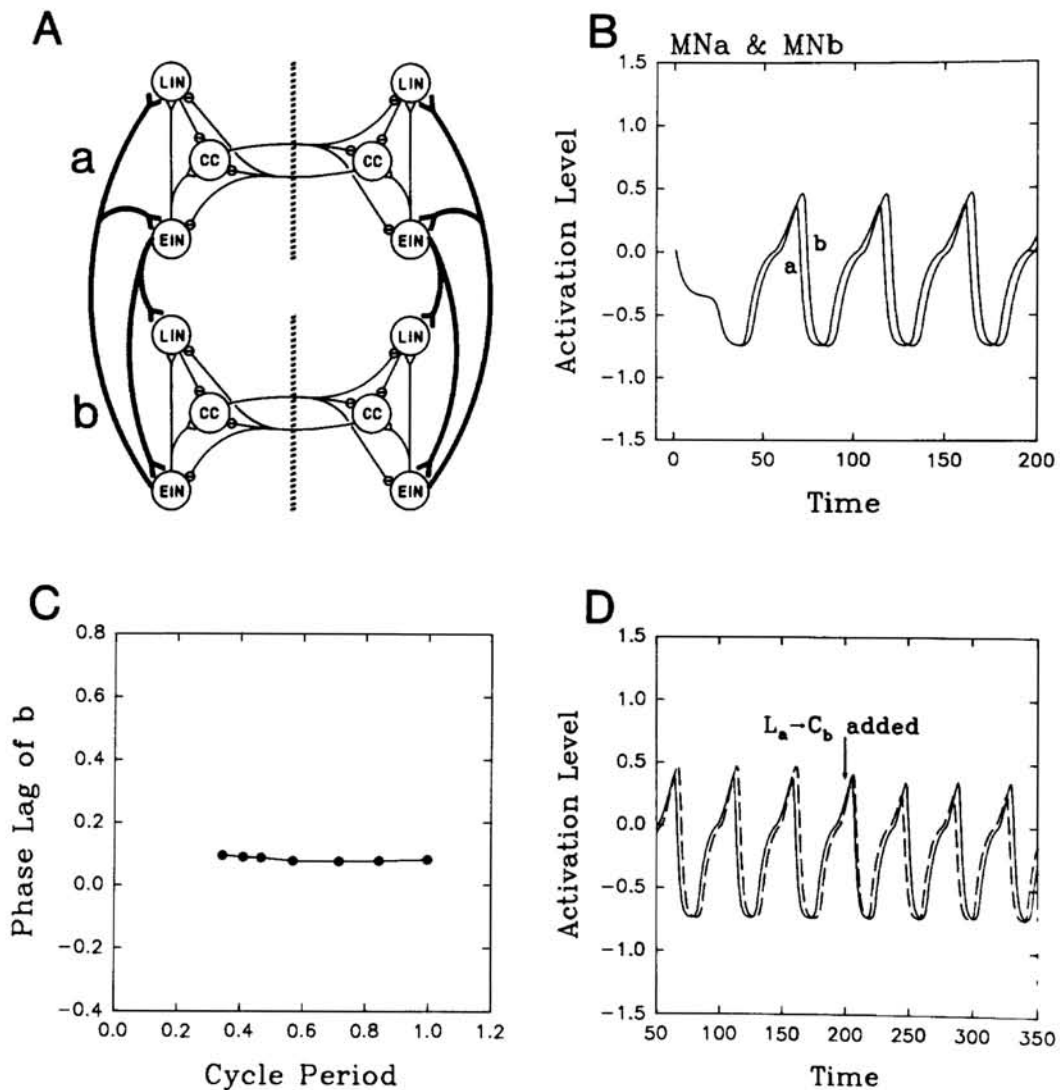

Figure 4: Coupling between two identical oscillators. **A,** the connectivity. **B,** steady-state coupling within a single cycle. **C,** constancy of phase lag over a range of oscillator periods. **D,** adding LIN→CC from oscillator a→b, reverses the phase, simulating backward swimming.

Each of the nine pairs of coupled interneurons between oscillators were capable of producing stable phase locking, although some coupling connections operated over a much wider range of synaptic weights than others. The steady-state phase difference between the oscillators and the rate of reaching it were also dependent on the synaptic weight of the coupling connections. The direction of the phase difference, that is, whether the postsynaptic oscillator was lagging or leading, depended both on the type of postsynaptic cell and the sign of the coupling input to it. If the postsynaptic cell was one which speeds the network (LIN or EIN) then their excitation by the coupling connection produced a lead of the postsynaptic network and their inhibition produced a lag. The opposite pattern held for CCs, which slow the network.

An example of a coupling scheme that satisfied several criteria for lamprey-like coupling is shown in Fig. 4. In this case (Fig. 4A), there was bidirectional, symmetric coupling of EINs in the two oscillators. This gave the network the ability to tolerate intrinsic frequency differences between the oscillators (buffering capacity). To provide a phase lag of oscillator b, EINs were connected to LINs bidirectionally but with greater weight in one direction (b→a). Such coupling reached a steady-state within a single cycle (Fig. 4B), and the phase difference was maintained at the same value over a range of cycle periods (Fig. 4C).

## 4.2  BACKWARD SWIMMING

It has been shown recently that there is rhythmic presynaptic inhibition of interneuronal axons in the lamprey spinal cord (Alford *et al.*, 1990). This type of cycle-by-cycle modulation of synaptic strength could account for shifts in phase coupling in the lamprey, such as occurs when the animal switches to brief bouts of backward swimming. One mechanism for backward swimming might be the inhibitory connection of LIN→CCs. The LINs have axons which descend up to 50 segments (one-half body length). In the neural network model, this descending inhibition of CC interneurons promotes backward swimming, *i.e.* a phase lead of the postsynaptic oscillators. Thus, presynaptic inhibition of these connections in nonlocal segments would allow forward swimming, while a removal of this presynaptic inhibition would initiate backward swimming (Fig. 4D).

## 5  CONCLUSIONS

The modeling described here demonstrates that the identified interneurons in the lamprey spinal cord may be multi-functional. They are known to contribute to the synaptic input to motoneurons during fictive swimming and thus to the shaping of the final motor output, but they may also function as components of the rhythm generating network itself. Finally, by virtue of their multi-segmental connections, they may have the additional role of providing the coupling signals among oscillators. Further experimental work will be required to determine which of these connections are actually used in the lamprey spinal cord for these functions.

## References

S. Alford, J. Christenson, & S. Grillner. (1990) Presynaptic $GABA_A$ and $GABA_B$ receptor-mediated phasic modulation in axons of spinal motor interneurons. *Eur. J. Neurosci.*, 3:107-117.

J.T. Buchanan. (1982) Identification of interneurons with contralateral, caudal axons in the lamprey spinal cord: synaptic interactions and morphology. *J. Neurophysiol.*, 47:961-975.

J.T. Buchanan. (1991) Electrophysiological properties of lamprey spinal neurons. *Soc. Neurosci. Abstr.*, 17:1581.

J.T. Buchanan. (1992) Neural network simulations of coupled locomotor oscillators in the lamprey spinal cord. *Biol. Cybern.*, 74: in press.

J.T. Buchanan & A.H. Cohen. (1982) Activities of identified interneurons, motoneurons, and muscle fibers during fictive swimming in the lamprey and effects of reticulospinal and dorsal cell stimulation. *J. Neurophysiol.*, 47:948-960.

J.T. Buchanan, S. Grillner, S. Cullheim, & M. Risling. (1989) Identification of excitatory interneurons contributing to generation of locomotion in lamprey: structure, pharmacology, and function. *J. Neurophysiol.*, 62:59-69.

A.H. Cohen. (1986) The intersegmental coordinating system of the lamprey: experimental and theoretical studies. In S. Grillner, P.S.G. Stein, D.G. Stuart, H. Forssberg, R.M. Herman (eds.), *Neurobiology of Vertebrate Locomotion*, 371-382. London: Macmillan.

A.H. Cohen & P. Wallen. (1980) The neuronal correlate of locomotion in fish: "fictive swimming" induced in an in vitro preparation of the lamprey spinal cord. *Exp. Brain Res.*, 41:11-18.

S. Grillner. (1981) Control of locomotion in bipeds, tetrapods, and fish. In V.B. Brooks (ed.), *Handbook of Physiology, Sect. 1. The Nervous System Vol. II. Motor Control*, 1179-1236. Maryland: Waverly Press.

J.A. Kahn. (1982) Patterns of synaptic inhibtion in motoneurons and interneurons during fictive swimming in the lamprey, as revealed by $Cl^-$ injections. *J. Comp. Neurol.*, 147:189-194.

C.M. Rovainen. (1974) Synaptic interactions of identified nerve cells in the spinal cord of the sea lamprey. *J. Comp. Neurol.*, 154:189-204.

D.F. Russell & P. Wallen. (1983) On the control of myotomal motoneurones during "fictive swimming" in the lamprey spinal cord in vitro. *Acta Physiol. Scand.*, 117:161-170.

A.I. Selverston, J.P. Miller, & M. Wadepuhl. (1983) Cooperative mechanisms for the production of rhythmic movements. *Sym. Soc. Exp. Biol.*, 37:55-88.

P. Wallen & T.L. Williams. (1984) Fictive locomotion in the lamprey spinal cord in vitro compared with swimming in the intact and spinal animal. *J. Physiol.*, 64:862-871.
